# An Approach to Bounded Rationality

**Eli Ben-Sasson**
Department of Computer Science
Technion — Israel Institute
of Technology

**Adam Tauman Kalai**
Department of Computer Science
College of Computing
Georgia Tech

**Ehud Kalai**
MEDS Department
Kellogg Graduate School of Management
Northwestern University

## Abstract

A central question in game theory and artificial intelligence is how a rational agent should behave in a complex environment, given that it cannot perform unbounded computations. We study strategic aspects of this question by formulating a simple model of a game with additional costs (computational or otherwise) for each strategy. First we connect this to zero-sum games, proving a counter-intuitive generalization of the classic min-max theorem to zero-sum games with the addition of strategy costs. We then show that potential games with strategy costs remain potential games. Both zero-sum and potential games with strategy costs maintain a very appealing property: simple learning dynamics converge to equilibrium.

## 1   The Approach and Basic Model

How should an intelligent agent play a complicated game like chess, given that it does not have unlimited time to think? This question reflects one fundamental aspect of "bounded rationality," a term coined by Herbert Simon [1]. However, bounded rationality has proven to be a slippery concept to formalize (prior work has focused largely on finite automata playing simple repeated games such as prisoner's dilemma, e.g. [2, 3, 4, 5]). This paper focuses on the *strategic* aspects of decision-making in complex multi-agent environments, i.e., on how a player should choose among strategies of varying complexity, given that its opponents are making similar decisions. Our model applies to general strategic games and allows for a variety of complexities that arise in real-world applications. For this reason, it is applicable to one-shot games, to extensive games, and to repeated games, and it generalizes existing models such as repeated games played by finite automata.

To easily see that bounded rationality can drastically affect the outcome of a game, consider the following *factoring game*. Player 1 chooses an $n$-bit number and sends it to Player 2, who attempts to find its prime factorization. If Player 2 is correct, he is paid 1 by Player 1, otherwise he pays 1 to Player 1. Ignoring complexity costs, the game is a trivial win for Player 2. However, for large $n$, the game should is essentially a win for Player 1, who can easily output a large random number that Player 2 cannot factor (under appropriate complexity assumptions).

In general, the outcome of a game (even a zero-sum game like chess) with bounded rationality is not so clear. To concretely model such games, we consider a set of *available* strategies along with *strategy costs*. Consider an example of two players preparing to play a computerized chess game for $100K prize. Suppose the players simultaneously choose among two available options: to use a $10K program A or an advanced program B, which costs $50K. We refer to the row chooser as white and to the column chooser as black, with the corresponding advantages reflected by the win

probabilities of white described in Table 1a. For example, when both players use program A, white wins 55% of the time and black wins 45% of the time (we ignore draws). The players naturally want to choose strategies to maximize their *expected net* payoffs, i.e., their expected payoff minus their cost. Each cell in Table 1b contains a pair of payoffs in units of thousands of dollars; the first is white's net expected payoff and the second is black's.

| a) | A | B |
|----|-----|-----|
| A | 55% | 13% |
| B | 93% | 51% |

| b) | A (-10) | B (-50) |
|---------|---------|---------|
| A (-10) | 45, 35 | **3, 37** |
| B (-50) | 43,-3 | 1,-1 |

Figure 1: a) Table of first-player winning probabilities based on program choices. b) Table of expected net earnings in thousands of dollars. The unique equilibrium is (A,B) which strongly favors the second player.

A surprising property is evident in the above game. Everything about the game seems to favor white. Yet due to the (symmetric) costs, at the unique Nash equilibrium (A,B) of Table 1b, black wins 87% of the time and nets $34K more than white. In fact, it is a dominant strategy for white to play A and for black to play B. To see this, note that playing B increases white's probability of winning by 38%, independent of what black chooses. Since the pot is $100K, this is worth $38K in expectation, but B costs $40K more than A. On the other hand, black enjoys a 42% increase in probability of winning due to B, independent of what white does, and hence is willing to pay the extra $40K.

Before formulating the general model, we comment on some important aspects of the chess example. First, traditional game theory states that chess can be solved in "only" two rounds of elimination of dominated strategies [10], and the outcome with optimal play should always be the same: either a win for white or a win for black. This theoretical prediction fails in practice: in top play, the outcome is very nondeterministic with white winning roughly twice as often as black. The game is too large and complex to be solved by brute force.

Second, we have been able to analyze the above chess program selection example exactly because we formulated as a game with a small number of available strategies per player. Another formulation that would fit into our model would be to include *all* strategies of chess, with some reasonable computational costs. However, it is beyond our means to analyze such a large game.

Third, in the example above we used monetary software cost to illustrate a type of strategy cost. But the same analysis could accommodate many other types of costs that can be measured numerically and subtracted from the payoffs, such as time or effort involved in the development or execution of a strategy, and other resource costs. Additional examples in this paper include the number of states in a finite automaton, the number of gates in a circuit, and the number of turns on a commuter's route. Our analysis is limited, however, to cost functions that depend only on the strategy of the player and not the strategy chosen by its opponent. For example, if our players above were renting computers A or B and paying for the time of actual usage, then the cost of using A would depend on the choice of computer made by the opponent.

Generalizing the example above, we consider a normal form game with the addition of strategy costs, a player-dependent cost for playing each available strategy. Our main results regard two important classes of games: constant-sum and potential games. Potential games with strategy costs remain potential games. While two-person constant-sum games are no longer constant, we give a basic structural description of optimal play in these games. Lastly, we show that known learning dynamics converge in both classes of games.

## 2 Definition of strategy costs

We first define an $N$-person normal-form game $G = (N, S, p)$ consisting of finite sets of (available) *pure strategies* $S = (S_1, \ldots, S_N)$ for the $N$ players, and a payoff function $p : S_1 \times \ldots \times S_N \rightarrow \mathbf{R}^N$. Players simultaneously choose strategies $s_i \in S_i$ after which player $i$ is rewarded with $p_i(s_1, \ldots, s_N)$. A randomized or *mixed strategy* $\sigma_i$ for player $i$ is a probability distribution over its pure strategies $S_i$,

$$\sigma_i \in \Delta_i = \left\{ x \in \mathbf{R}^{|S_i|} : \sum x_j = 1, x_j \geq 0 \right\}.$$

We extend $p$ to $\Delta_1 \times \ldots \times \Delta_N$ in the natural way, i.e., $p_i(\sigma_1, \ldots, \sigma_N) = \mathbf{E}[p_i(s_1, \ldots, s_N)]$ where each $s_i$ is drawn from $\sigma_i$, independently. Denote by $s_{-i} = (s_1, s_2, \ldots, s_{i-1}, s_{i+1}, \ldots, s_N)$ and similarly for $\sigma_{-i}$. A *best response* by player $i$ to $\sigma_{-i}$ is $\sigma_i \in \Delta_i$ such that $p_i(\sigma_i, \sigma_{-i}) = \max_{\sigma_i' \in \Delta_i} p_i(\sigma_i', \sigma_{-i})$. A (mixed strategy) *Nash equilibrium* of $G$ is a vector of strategies $(\sigma_1, \ldots, \sigma_N) \in \Delta_1 \times \ldots \times \Delta_N$ such that each $\sigma_i$ is a best response to $\sigma_{-i}$.

We now define $G^{-c}$, the game $G$ with strategy costs $c = (c_1, \ldots, c_N)$, where $c_i : S_i \to \mathbf{R}$. It is simply an $N$-person normal-form game $G^{-c} = (N, S, p^{-c})$ with the same sets of pure strategies as $G$, but with a new payoff function $p^{-c} : S_1 \times \ldots \times S_N \to \mathbf{R}^N$ where,

$$p_i^{-c}(s_1, \ldots, s_N) = p_i(s_1, \ldots, s_N) - c_i(s_i), \text{for } i = 1, \ldots, N.$$

We similarly extend $c_i$ to $\Delta_i$ in the natural way.

# 3  Two-person constant-sum games with strategy costs

Recall that a game is constant-sum ($k$-sum for short) if at every combination of individual strategies, the players' payoffs sum to some constant k. Two-person $k$-sum games have some important properties, not shared by general sum games, which result in more effective game-theoretic analysis.

In particular, every $k$-sum game has a unique *value* $v \in \mathbf{R}$. A mixed strategy for player 1 is called *optimal* if it guarantees payoff $\geq v$ against any strategy of player 2. A mixed strategy for player 2 is *optimal* if it guarantees $\geq k - v$ against any strategy of player 1. The term *optimal* is used because optimal strategies guarantee as much as possible ($v + k - v = k$) and playing anything that is not optimal can result in a lesser payoff, if the opponent responds appropriately. (This fact is easily illustrated in the game rock-paper-scissors – randomizing uniformly among the strategies guarantees each player 50% of the pot, while playing anything other than uniformly random enables the opponent to win strictly more often.) The existence of optimal strategies for both players follows from the min-max theorem. An easy corollary is that the Nash equilibria of a $k$-sum game are *exchangeable*: they are simply the cross-product of the sets of optimal mixed strategies for both players. Lastly, it is well-known that equilibria in two-person $k$-sum games can be *learned* in repeated play by simple dynamics that are guaranteed to converge [17].

With the addition of strategy costs, a $k$-sum game is no longer $k$-sum and hence it is not clear, at first, what optimal strategies there are, if any. (Many examples of general-sum games do not have optimal strategies.) We show the following generalization of the above properties for zero-sum games with strategies costs.

**Theorem 1.** *Let $G$ be a finite two-person $k$-sum game and $G^{-c}$ be the game with strategy costs $c = (c_1, c_2)$.*

1. *There is a value $v \in \mathbf{R}$ for $G^{-c}$ and nonempty sets $OPT_1$ and $OPT_2$ of optimal mixed strategies for the two players. $OPT_1$ is the set of strategies that guarantee player 1 payoff $\geq v - c_2(\sigma_2)$, against any strategy $\sigma_2$ chosen by player 2. Similarly, $OPT_2$ is the set of strategies that guarantee player 2 payoff $\geq k - v - c_1(\sigma_1)$ against any $\sigma_1$.*

2. *The Nash equilibria of $G^{-c}$ are exchangeable: the set of Nash equilibria is $OPT_1 \times OPT_2$.*

3. *The set of net payoffs possible at equilibrium is an axis-parallel rectangle in $\mathbf{R}^2$.*

For zero-sum games, the term optimal strategy was natural: the players could guarantee $v$ and $k - v$, respectively, and this is all that there was to share. Moreover, it is easy to see that only pairs of optimal strategies can have the Nash equilibria property, being best responses to each other.

In the case of zero-sum games with strategy costs, the optimal structure is somewhat counterintuitive. First, it is strange that the amount guaranteed by either player depends on the cost of the other player's action, when in reality each player pays the cost of its own action. Second, it is not even clear why we call these optimal strategies. To get a feel for this latter issue, notice that the sum of the net payoffs to the two players is always $k - c_1(\sigma_1) - c_2(\sigma_2)$, which is exactly the total of what optimal strategies guarantee, $v - c_2(\sigma_2) + k - v - c_1(\sigma_1)$. Hence, if both players play what we call optimal strategies, then neither player can improve and they are at Nash equilibrium. On the other hand, suppose player 1 selects a strategy $\sigma_1$ that does not guarantee him payoff at least

$v - c_2(\sigma_2)$. This means that there is some response $\sigma_2$ by player 2 for which player 1's payoff is $< v - c_2(\sigma_2)$ and hence player 2's payoff is $> k - v - c_1(\sigma_1)$. Thus player 2's best response to $\sigma_1$ must give player 2 payoff $> k - v - c_1(\sigma_1)$ and leave player 1 with $< v - c_2(\sigma_2)$.

The proof of the theorem (the above reasoning only implies part 2 from part 1) is based on the following simple observation. Consider the $k$-sum game $H = (N, S, q)$ with the following payoffs:

$$q_1(s_1, s_2) = p_1(s_1, s_2) - c_1(s_1) + c_2(s_2) = p_1^{-c}(s_1, s_2) + c_2(s_2)$$

$$q_2(s_1, s_2) = p_2(s_1, s_2) - c_2(s_1) + c_1(s_1) = p_2^{-c}(s_1, s_2) + c_1(s_1)$$

That is to say, Player 1 pays its strategy cost to Player 2 and vice versa. It is easy to verify that,

$$\forall \sigma_1, \sigma_1' \in \Delta_1, \sigma_2 \in \Delta_2 \quad q_1(\sigma_1, \sigma_2) - q_1(\sigma_1', \sigma_2) = p_1^{-c}(\sigma_1, \sigma_2) - p_1^{-c}(\sigma_1', \sigma_2) \qquad (1)$$

This means that the relative advantage in switching strategies in games $G^{-c}$ and $H$ are the same. In particular, $\sigma_1$ is a best response to $\sigma_2$ in $G^{-c}$ if and only if it is in $H$. A similar equality holds for player 2's payoffs. Note that these conditions imply that the games $G^{-c}$ and $H$ are *strategically equivalent* in the sense defined by Moulin and Vial [16].

*Proof of Theorem 1.* Let $v$ be the value of the game $H$. For any strategy $\sigma_1$ that guarantees player 1 payoff $\geq v$ in $H$, $\sigma_1$ guarantees player 1 $\geq v - c_2(\sigma_2)$ in $G^{-c}$. This follows from the definition of $H$. Similarly, any strategy $\sigma_2$ that guarantees player 2 payoff $\geq k - v$ in $H$ will guarantee $\geq k - v - c_1(\sigma_1)$ in $G^{-c}$. Thus the sets $\text{OPT}_1$ and $\text{OPT}_2$ are non-empty. Since $v - c_2(\sigma_2) + k - v - c_1(\sigma_1) = k - c_1(\sigma_1) - c_2(\sigma_2)$ is the sum of the payoffs in $G^{-c}$, nothing greater can be guaranteed by either player.

Since the best responses of $G^{-c}$ and $H$ are the same, the Nash equilibria of the two games are the same. Since H is a $k$-sum game, its Nash equilibria are exchangeable, and thus we have part 2. (This holds for any game that is strategically equivalent to $k$-sum.)

Finally, the optimal mixed strategies $\text{OPT}_1$, $\text{OPT}_2$ of any $k$-sum game are convex sets. If we look at the achievable costs of the mixed strategies in $\text{OPT}_i$, by the definition of the cost of a mixed strategy, this will be a convex subset of $\mathbf{R}$, i.e., an interval. By parts 1 and 2, the set of achievable net payoffs at equilibria of $G^{-c}$ are therefore the cross-product of intervals. $\square$

To illustrate Theorem 1 graphically, Figure 2 gives a $4 \times 4$ example with costs of 1, 2, 3, and 4, respectively. It illustrates a situation with multiple optimal strategies. Notice that player 1 is completely indifferent between its optimal choices A and B, and player 2 is completely indifferent between C and D. Thus the only question is how kind they would like to be to their opponent. The (A,C) equilibrium is perhaps most natural as it is yields the highest payoffs for both parties.

Note that the proof of the above theorem actually shows that zero-sum games with costs share additional appealing properties of zero-sum games. For example, computing optimal strategies is a polynomial time-computation in an $n \times n$ game, as it amounts to computing the equilibria of $H$. We next show that they also have appealing learning properties, though they do not share all properties of zero-sum games.[1]

## 3.1 Learning in repeated two-person $k$-sum games with strategy costs

Another desirable property of $k$-sum games is that, in repeated play, natural learning dynamics converge to the set of Nash equilibria. Before we state the analogous conditions for $k$-sum games with costs, we briefly give a few definitions. A *repeated game* is one in which players chooses a sequence of strategies vectors $s^1, s^2, \ldots$, where each $s^t = (s_1^t, \ldots, s_N^t)$ is a strategy vector of some fixed *stage game* $G = (N, S, p)$. Under perfect monitoring, when selecting an action in any period the players know all the previous selected actions. As we shall discuss, it is possible to learn to play without perfect monitoring as well.

| a) | A | B | C | D |
|---|---|---|---|---|
| A | 6, 4 | 5, 5 | 3, 7 | 2, 8 |
| B | 7, 3 | 6, 4 | 4, 6 | 3, 7 |
| C | 7.5, 2.5 | 6.5, 3.5 | 4.5, 5.5 | 3.5, 6.5 |
| D | 8.5, 1.5 | 7, 3 | 5.5, 4.5 | 4.5, 5.5 |

| b) | A (-1) | B (-2) | C (-3) | D (-4) |
|---|---|---|---|---|
| A (-1) | 5, 3 | 4, 3 | **2, 4** | **1, 4** |
| B (-2) | 5, 2 | 4, 2 | **2, 3** | **1, 3** |
| C (-3) | 4.5, 1.5 | 3.5, 1.5 | 1.5, 2.5 | 0.5, 2.5 |
| D (-4) | 4.5, 0.5 | 3, 1 | 1.5, 1.5 | 0.5, 1.5 |

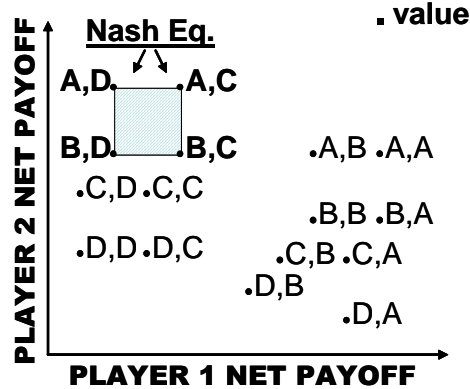

Figure 2: a) Payoffs in 10-sum game $G$. b) Expected net earnings in $G^{-c}$. $OPT_1$ is any mixture of A and B, and $OPT_2$ is any mixture of C and D. Each player's choice of equilibrium strategy affects only the opponent's net payoff. c) A graphical display of the payoff pairs. The shaded region shows the rectangular set of payoffs achievable at mixed strategy Nash equilibria.

Perhaps the most intuitive dynamics are best-response: at each stage, each player selects a best response to the opponent's previous stage play. Unfortunately, these naive dynamics fails to converge to equilibrium in very simple examples. The *fictitious play* dynamics prescribe, at stage $t$, selecting any strategy that is a best response to the empirical distribution of opponent's play during the first $t-1$ stages. It has been shown that fictitious play converges to equilibrium (of the stage game $G$) in $k$-sum games [17].

However, fictitious play requires perfect monitoring. One can learn to play a two-person $k$-sum game with no knowledge of the payoff table or anything about the other players actions. Using experimentation, the only observations required by each player are its own payoffs in each period (in addition to the number of available actions). So-called *bandit algorithms* [7] must manage the exploration-exploitation tradeoff. The proof of their convergence follows from the fact that they are *no-regret* algorithms. (No-regret algorithms date back to Hannan in the 1950's [12], but his required perfect monitoring). The *regret* of a player $i$ at stage $T$ is defined to be,

$$\text{regret of } i \text{ at } T = \frac{1}{T} \max_{s_i \in S_i} \sum_{t=1}^{T} \left( p_i(s_i, s_{-i}^t) - p_i(s_i^t, s_{-i}^t) \right),$$

that is, how much better in hindsight player $i$ could have done on the first $T$ stages had it used one fixed strategy the whole time (and had the opponents not changed their strategies). Note that regret can be positive or negative. A no-regret algorithm is one in which each player's asymptotic regret converges to $(-\infty, 0]$, i.e., is guaranteed to approach 0 or less. It is well-known that no-regret condition in two-person $k$-sum games implies convergence to equilibrium (see, e.g., [13]). In particular, the pair of mixed strategies which are the empirical distributions of play over time approaches the set of Nash equilibrium of the stage game.

Inverse-polynomial rates of convergence (that are polynomial also in the size of the game) can be given for such algorithms. Hence no-regret algorithms provide arguably reasonable ways to play a $k$-sum game of moderate size. Note that in general-sum games, no such dynamics are known. Fortunately, the same algorithm that works for learning in $k$-sum games seem to work for learning in such games with strategy costs.

**Theorem 2.** *Fictitious play converges to the set of Nash equilibria of the stage game in a two-person $k$-sum game with strategy costs, as do no-regret learning dynamics.*

*Proof.* The proof again follows from equation (1) regarding the game $H$. Fictitious play dynamics are defined only in terms of best response play. Since $G^{-c}$ and $H$ share the same best responses, fictitious play dynamics are identical for the two games. Since they share the same equilibria and fictitious play converges to equilibria in $H$, it must converge in $G^{-c}$ as well.

For no-regret algorithms, equation (1) again implies that for any play sequence, the regret of each player $i$ with respect to game $G^{-c}$ is the same as its regret with respect to the game $H$. Hence, no regret in $G^{-c}$ implies no regret in $H$. Since no-regret algorithms converge to the set of equilibria in $k$-sum games, they converge to the set of equilibria in $H$ and therefore in $G^{-c}$ as well. □

## 4 Potential games with strategic costs

Let us begin with an example of a *potential game*, called a *routing game* [18]. There is a fixed directed graph with $n$ nodes and $m$ edges. Commuters $i = 1, 2, \ldots, N$ each decide on a route $\pi_i$, to take from their home $s_i$ to their work $t_i$, where $s_i$ and $t_i$ are nodes in the graph. For each edge, $uv$, let $n_{uv}$ be the number of commuters whose path $\pi_i$ contains edge $uv$. Let $f_{uv} : \mathbb{Z} \to \mathbf{R}$ be a non-negative monotonically increasing congestion function. Player $i$'s payoff is $-\sum_{uv \in \pi_i} f_{uv}(n_{uv})$, i.e., the negative sum of the congestions on the edges in its path.

An $N$-person normal form game $G$ is said to be a potential game [15] if there is some potential function $\Phi : S_1 \times \ldots S_N \to \mathbf{R}$ such that changing a single player's action changes its payoff by the change in the potential function. That is, there exists a single function $\Phi$, such that for all players $i$ and all pure strategy vectors $s, s' \in S_1 \times \ldots \times S_N$ that differ only in the $i$th coordinate,

$$p_i(s) - p_i(s') = \Phi(s) - \Phi(s'). \tag{2}$$

Potential games have appealing learning properties: simple better-reply dynamics converge to pure-strategy Nash equilibria, as do the more sophisticated fictitious-play dynamics described earlier [15]. In our example, this means that if players change their individual paths so as to selfishly reduce the sum of congestions on their path, this will eventually lead to an equilibrium where no one can improve. (This is easy to see because $\Phi$ keeps increasing.) The absence of similar learning properties for general games presents a frustrating hole in learning and game theory.

It is clear that the theoretically clean commuting example above misses some realistic considerations. One issue regarding complexity is that most commuters would not be willing to take a very complicated route just to save a short amount of time. To model this, we consider potential games with strategy costs. In our example, this would be a cost associated with every path. For example, suppose the graph represented streets in a given city. We consider a natural strategy complexity cost associated with a route $\pi$, say $\lambda(\#\text{turns}(\pi))^2$, where there is a parameter $\lambda \in \mathbf{R}$ and $\#\text{turns}(\pi)$ is defined as the number of times that a commuter has to turn on a route. (To be more precise, say each edge in the graph is annotated with a street name, and a turn is defined to be a pair of consecutive edges in the graph with different street names.) Hence, a best response for player $i$ would minimize:

$$\min_{\pi \text{ from } s_i \text{ to } t_i} (\text{total congestion of } \pi) + \lambda(\#\text{turns}(\pi))^2.$$

While adding strategy costs to potential games allows for much more flexibility in model design, one might worry that appealing properties of potential games, such as having pure strategy equilibria and easy learning dynamics, no longer hold. This is not the case. We show that strategic costs fit easily into the potential game framework:

**Theorem 3.** *For any potential game $G$ and any cost functions $c$, $G^{-c}$ is also a potential game.*

*Proof.* Let $\Phi$ be a potential function for $G$. It is straightforward to verify that the $G^{-c}$ admits the following potential function $\Phi'$:

$$\Phi'(s_1, \ldots, s_N) = \Phi(s_1, \ldots, s_N) - c_1(s_1) - \ldots - c_N(s_N). \qquad \square$$

## 5  Additional remarks

Part of the reason that the notion of bounded rationality is so difficult to formalize is that understanding enormous games like chess is a daunting proposition. That is why we have narrowed it down to choosing among a small number of available programs.

A game theorist might begin by examining the *complete* payoff table of Figure 1a, which is prohibitively large. Instead of considering only the choices of programs A and B, each player considers all possible chess strategies. In that sense, our payoff table in 1a would be viewed as a reduction of the "real" normal form game. A computer scientist, on the other hand, may consider it reasonable to begin with the existing strategies that one has access to. Regardless of how you view the process, it is clear that for practical purposes players in real life do simplify and analyze "smaller" sets of strategies. Even if the players consider the option of engineering new chess-playing software, this can be viewed as a third strategy in the game, with its own cost and expected payoffs.

Again, when considering small number of available strategies, like the two programs above, it may still be difficult to assess the expected payoffs that result when (possibly randomized) strategies play against each other. An additional assumption made throughout the paper is that the players share the same assessments about these expected payoffs. Like other common-knowledge assumptions made in game theory, it would be desirable to weaken this assumption. In the special families of games studied in this paper, and perhaps in additional cases, learning algorithms may be employed to reach equilibrium without knowledge of payoffs.

### 5.1  Finite automata playing repeated games

There has been a large body of interesting work on repeated games played by finite automata (see [14] for a survey). Much of this work is on achieving cooperation in the classic prisoner's dilemma game (e.g., [2, 3, 4, 5]). Many of these models can be incorporated into the general model outlined in this paper.

For example, to view the Abreu and Rubinstein model [6] as such, consider the normal form of an infinitely repeated game with discounting, but restricted to strategies that can be described by finite automata (the payoffs in every cell of the payoff table are the discounted sums of the infinite streams of payoffs obtained in the repeated game). Let the cost of a strategy be an increasing function of the number of states it employs.

For Neyman's model [3], consider the normal form of a finitely repeated game with a known number of repetitions. You may consider strategies in this normal form to be only ones with a bounded number of states, as required by Neyman, and assign zero cost to all strategies. Alternatively, you may allow all strategies but assign zero cost to ones that employ number of states below Neyman's bounds, and an infinite cost to strategies that employ a number of states that exceeds Neyman's bounds.

The structure of equilibria proven in Theorem 1 applies to all the above models when dealing with repeated $k$-sum games, as in [2].

## 6  Future work

There are very interesting questions to answer about bounded rationality in truly large games that we did not touch upon. For example, consider the factoring game from the introduction. A pure strategy for Player 1 would be outputting a single $n$-bit number. A pure strategy for Player 2 would be any factoring program, described by a circuit that takes as input an $n$-bit number and attempts to output a representation of its prime factorization. The complexity of such a strategy would be an increasing function of the number of gates in the circuit. It would be interesting to make connections between asymptotic algorithm complexity and games.

Another direction regards an elegant line of work on learning to play correlated equilibria by repeated play [11]. It would be natural to consider how strategy costs affect correlated equilibria. Finally, it would also be interesting to see how strategy costs affect the so-called "price of anarchy" [19] in congestion games.

**Acknowledgments**

This work was funded in part by a U.S. NSF grant SES-0527656, a Landau Fellowship supported by the Taub and Shalom Foundations, a European Community International Reintegration Grant, an Alon Fellowship, ISF grant 679/06, and BSF grant 2004092. Part of this work was done while the first and second authors were at the Toyota Technological Institute at Chicago.

## Footnotes

[1] One property that is violated by the chess example is the "advantage of an advantage" property. Say Player 1 *has the advantage* over Player 2 in a square game if $p_1(s_1, s_2) \geq p_2(s_2, s_1)$ for all strategies $s_1, s_2$. At equilibrium of a $k$-sum game, a player with the advantage must have a payoff at least as large as its opponent. This is no longer the case after incorporating strategy costs, as seen in the chess example, where Player 1 has the advantage (even including strategy costs), yet his equilibrium payoff is smaller than 2's.

## References

[1] H. Simon. *The sciences of the artificial*. MIT Press, Cambridge, MA, 1969.

[2] E. Ben-Porath. Repeated games with finite automata, *Journal of Economic Theory* 59: 17–32, 1993.

[3] A. Neyman. Bounded Complexity Justifies Cooperation in the Finitely Repeated Prisoner's Dilemma. *Economic Letters*, 19: 227–229, 1985.

[4] A. Rubenstein. Finite automata play the repeated prisoner's dilemma. *Journal of Economic Theory*, 39:83–96, 1986.

[5] C. Papadimitriou, M. Yannakakis: On complexity as bounded rationality. In *Proceedings of the Twenty-Sixth Annual ACM Symposium on Theory of Computing*, pp. 726–733, 1994.

[6] D. Abreu and A. Rubenstein. The Structure of Nash Equilibrium in Repeated Games with Finite Automata. *Econometrica* 56:1259-1281, 1988.

[7] P. Auer, N. Cesa-Bianchi, Y. Freund, R. Schapire. The Nonstochastic Multiarmed Bandit Problem. *SIAM J. Comput.* 32(1):48-77, 2002.

[8] X. Chen, X. Deng, and S. Teng. Computing Nash Equilibria: Approximation and smoothed complexity. Electronic Colloquium on Computational Complexity Report TR06-023, 2006.

[9] K. Daskalakis, P. Goldberg, C. Papadimitriou. The complexity of computing a Nash equilibrium. Electronic Colloquium on Computational Complexity Report TR05-115, 2005.

[10] C. Ewerhart. Chess-like Games Are Dominance Solvable in at Most Two Steps. *Games and Economic Behavior*, 33:41-47, 2000.

[11] D. Foster and R. Vohra. Regret in the on-line decision problem. *Games and Economic Behavior*, 21:40-55, 1997.

[12] J. Hannan. Approximation to Bayes risk in repeated play. In M. Dresher, A. Tucker, and P. Wolfe, editors, *Contributions to the Theory of Games*, volume 3, pp. 97–139. Princeton University Press, 1957.

[13] S. Hart and A. Mas-Colell. A General Class of Adaptive Strategies. *Journal of Economic Theory* 98(1):26–54, 2001.

[14] E. Kalai. Bounded rationality and strategic complexity in repeated games. In T. Ichiishi, A. Neyman, and Y. Tauman, editors, *Game Theory and Applications*, pp. 131–157. Academic Press, San Diego, 1990.

[15] D. Monderer, L. Shapley. Potential games. *Games and Economic Behavior*, 14:124–143, 1996.

[16] H. Moulin and P. Vial. Strategically Zero Sum Games: the Class of Games Whose Completely Mixed Equilibria Cannot Be Improved Upon. *International Journal of Game Theory*, 7:201–221, 1978.

[17] J. Robinson, An iterative method of solving a game, *Ann. Math.* 54:296–301, 1951.

[18] R. Rosenthal. A Class of Games Possessing Pure-Strategy Nash Equilibria. *International Journal of Game Theory*, 2:65–67, 1973.

[19] E. Koutsoupias and C. Papadimitriou. Worstcase equilibria. In *Proceedings of the 16th Annual Symposium on Theoretical Aspects of Computer Science*, pp. 404–413, 1999.